# Analysis of Brain States
# from Multi-Region LFP Time-Series

**Kyle Ulrich** [1], **David E. Carlson** [1], **Wenzhao Lian** [1], **Jana Schaich Borg** [2], **Kafui Dzirasa** [2] **and Lawrence Carin** [1]

[1] Department of Electrical and Computer Engineering
[2] Department of Psychiatry and Behavioral Sciences
Duke University, Durham, NC 27708
{kyle.ulrich, david.carlson, wenzhao.lian, jana.borg,
kafui.dzirasa, lcarin}@duke.edu

## Abstract

The local field potential (LFP) is a source of information about the broad patterns of brain activity, and the frequencies present in these time-series measurements are often highly correlated between regions. It is believed that these regions may jointly constitute a "brain state," relating to cognition and behavior. An infinite hidden Markov model (iHMM) is proposed to model the evolution of brain states, based on electrophysiological LFP data measured at multiple brain regions. A brain state influences the spectral content of each region in the measured LFP. A new state-dependent tensor factorization is employed across brain regions, and the spectral properties of the LFPs are characterized in terms of Gaussian processes (GPs). The LFPs are modeled as a mixture of GPs, with state- and region-dependent mixture weights, and with the spectral content of the data encoded in GP spectral mixture covariance kernels. The model is able to estimate the number of brain states and the number of mixture components in the mixture of GPs. A new variational Bayesian split-merge algorithm is employed for inference. The model infers state changes as a function of external covariates in two novel electrophysiological datasets, using LFP data recorded simultaneously from multiple brain regions in mice; the results are validated and interpreted by subject-matter experts.

## 1   Introduction

Neuroscience has made significant progress in learning how activity in specific neurons or brain areas correlates with behavior. One of the remaining mysteries is how to best represent and understand the way whole-brain activity relates to cognition: in other words, how to describe brain states [1]. Although different brain regions have different functions, neural activity across brain regions is often highly correlated. It has been proposed that the specific way brain regions are correlated at any given time may represent a "state" designed specifically to optimize neural computations relevant to the behavioral context an organism is in [2]. Unfortunately, although there is great interest in the concept of global brain states, little progress has been made towards developing methods to identify or characterize them.

The study of arousal is an important area of research relating to brain states. Arousal is a hotly debated topic that generally refers to the way the brain dynamically responds to varying levels of stimulation [3]. One continuum of arousal used in the neuroscience literature is sleep (low arousal) to wakefulness (higher arousal). Another is calm (low arousal) to excited or stressed (high arousal) [4]. A common electrophysiological measurement used to determine arousal levels is local field potentials (LFPs), or low-frequency ($< 200\,Hz$) extracellular neural oscillations that represent coordinated

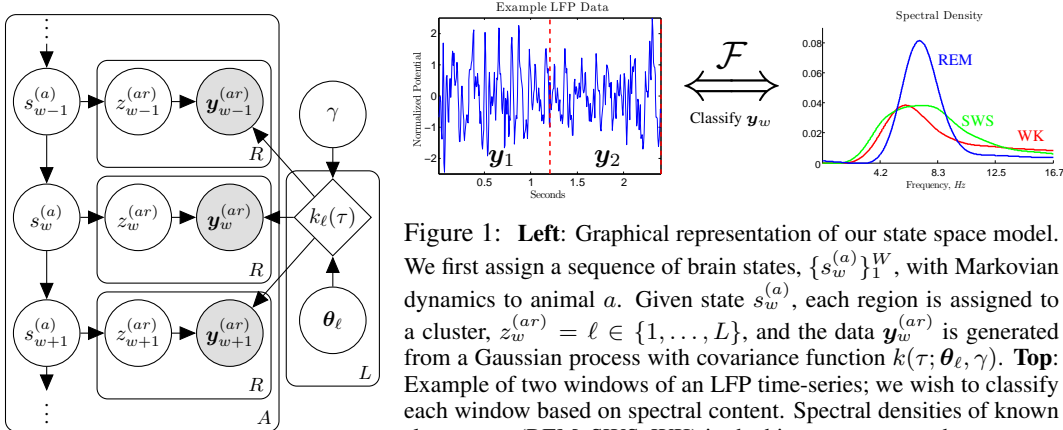

Figure 1: **Left**: Graphical representation of our state space model. We first assign a sequence of brain states, $\{s_w^{(a)}\}_1^W$, with Markovian dynamics to animal $a$. Given state $s_w^{(a)}$, each region is assigned to a cluster, $z_w^{(ar)} = \ell \in \{1, \ldots, L\}$, and the data $\boldsymbol{y}_w^{(ar)}$ is generated from a Gaussian process with covariance function $k(\tau; \boldsymbol{\theta}_\ell, \gamma)$. **Top**: Example of two windows of an LFP time-series; we wish to classify each window based on spectral content. Spectral densities of known sleep states (REM, SWS, WK) in the hippocampus are shown.

neural activity across distributed spatial and temporal scales. LFPs are useful for describing overall brain states since they reflect activity across many neural networks. We examine brain states under different levels of arousal by recording LFPs simultaneously in multiple regions of the mouse brain, first, as mice pass through different stages of sleep, and second, as mice are moved from a familiar environment to a novel environment to induce interest and exploration.

In neuroscience, the analysis of electrophysiological time-series data is largely centered around dynamic causal modeling (DCM) [5], where continuous state-space models are formulated based on differential equations that are specifically crafted around knowledge of underlying neurobiological processes. However, DCM is not suitable for exploratory analysis of data, such as inferring unknown arousal levels, for two reasons: because the differential equations are driven by inputs of experimental conditions, and the analysis is dependent on *a priori* hypotheses about which neuronal populations and interactions are important. This work focuses on methods suitable for exploratory analysis.

Previously published neuroscience studies distinguished between slow-wave sleep (SWS), rapid-eye-movement (REM), and wake (WK) using proportions of high-frequency (33-55 Hz) gamma oscillations and lower frequency theta (4-9 Hz) oscillations in a brain area called the Hippocampus [6, 7]. As an alternative approach, recent statistical methods for tensor factorization [8] can be applied to short time Fourier transform (STFT) coefficients by factorizing a 3–way LFP tensor, with dimensions of brain region, frequency band and time. Distinct sleep states may then be revealed by clustering the inferred sequence of time-varying score vectors.

Although good first steps, the above two methods have several shortcomings: 1) They do not consider the time dependency of brain activity, and therefore cannot capture state-transition properties. 2) They cannot directly work on raw data, but require preprocessing that only considers spectral content in predefined frequency bins, thus leading to information loss. 3) They do not allow for individual brain regions to take on their own set of sub-state characteristics within a given global brain state. 4) Finally, they cannot leverage the shared information of LFP data across multiple animals.

In this paper we overcome the shortcomings of previously published brain-state methods by defining a sequence of brain states over a sliding window of raw, filtered LFP data, where we impose an infinite hidden Markov model (iHMM) [9] on these state assignments. Conditioned on this brain state, each brain region is assigned to a cluster in a mixture model. Each cluster is associated with a specific spectral content (or density) pattern, manifested through a spectral mixture kernel [10] of a Gaussian process. Each window of LFP data is generated as a draw from this mixture of Gaussian processes. Thus, all animals share an underlying brain state space, of which, all brain regions share the underlying components of the mixture model.

## 2   Model

For each animal $a \in \{1, \ldots A\}$, we have time-series of the LFP in $R$ different regions, measured simultaneously. These time-series are split into sequential, sliding *windows*, $\boldsymbol{y}_w^{(ar)} \in \mathbb{R}^N$ for $w \in \{1, \ldots, W\}$, such that windows are common across regions. These windows are chosen to be overlapping, thereby sharing data points between consecutive windows; nonoverlapping win-

dows may also be used. Each window is considered as a single observation vector, and we wish to model the generative process of these observations, $\{\boldsymbol{y}_w^{(ar)}\}$.

The proposed model aims to describe the spectral content in each of these LFP signals, as a function of brain region and time. This is done by first assigning a *joint* "brain state" to each time window, $\{s_1^{(a)}, \ldots, s_W^{(a)}\}$, shared across all brain regions $\{1, \ldots, R\}$. The brain state is assumed to evolve in time as a latent Markov process. The LFP data from a particular brain region is assumed drawn from a mixture of Gaussian processes. The characteristics of each mixture component are shared across brain states and brain regions, with mixture weights that are dependent on these two entities.

## 2.1 Brain state assignment

Within the generative process, each animal has a latent *brain state* for every time window, $w$. This brain state is represented through a categorical latent variable $s_w^{(a)}$, and an infinite hidden Markov model (iHMM) is placed on the state dynamics [9, 12]. This process is formulated as

$$s_w^{(a)} \sim \text{Categorical}(\boldsymbol{\lambda}_{s_{w-1}^{(a)}}^{(a)}), \qquad \boldsymbol{\lambda}_g^{(a)} \sim \text{DP}(\alpha_0 \boldsymbol{\beta}), \qquad \boldsymbol{\beta} \sim \text{GEM}(\gamma_0), \qquad (1)$$

where GEM is the stick-breaking process $\beta_h = \beta_h' \prod_{i=1}^{h-1}(1 - \beta_i')$ with $\beta_h' \sim \text{Beta}(1, \gamma_0)$. Here, $\{\beta_h\}_{h=1}^H$ represents global transition probabilities to each state in a potentially infinite state space. For the stick-breaking process, $H \to \infty$, but in a finite collection of data only a finite number of state transitions will be used and $H$ can be efficiently truncated. Since the state space is shared across animals, we cannot predefine initial state assignments, $s_1^{(a)}$. To remedy this, we allow $s_1^{(a)} \sim \text{Categorical}(\boldsymbol{\psi}^{(a)})$ and place a discrete uniform prior on $\boldsymbol{\psi}^{(a)}$ over the truncated state space.

Each animal is given a transition matrix $\boldsymbol{\Lambda}^{(a)}$, where each row of this matrix is a transition probability vector $\boldsymbol{\lambda}_g^{(a)}$ such that the transition from state $g$ to state $h$ for animal $a$ is $\lambda_{gh}^{(a)}$, each centered around the global transition vector $\boldsymbol{\beta}$. Because each animal's brain can be structured differently (e.g., as an extreme case, consider a central nervous system disorder), we allow $\boldsymbol{\Lambda}^{(a)}$ to vary from animal to animal.

## 2.2 Assigning brain regions to clusters

For each brain state, mixture weights are drawn to define the distribution over *clusters* independently for each region $r$, centered around a global mixture $\boldsymbol{\eta}$ using a hierarchical Dirichlet Process [12]:

$$\boldsymbol{\phi}_h^{(r)} \sim \text{DP}(\alpha_1 \boldsymbol{\eta}), \qquad \boldsymbol{\eta} \sim \text{GEM}(\gamma_1), \qquad (2)$$

where $\phi_{h\ell}^{(r)}$ is the probability of assigning region $r$ of a window with brain state $h$ to cluster $\ell$. This cluster assignment can be written as

$$z_w^{(ar)} | s_w^{(a)} \sim \text{Categorical}(\boldsymbol{\phi}_{s_w^{(a)}}^{(r)}). \qquad (3)$$

For each cluster $\ell$ there is a set of parameters, $\boldsymbol{\theta}_\ell$, describing a Gaussian process (GP), detailed in Section 2.3. One could consider the joint probability over cluster assignments for all brain regions as an extension of a latent nonnegative PARAFAC tensor decomposition [11, 13]. We refer to the Supplemental Material for details. Our clustering model differs from the infinite tensor factorization (ITF) model of [11] in three significant ways: we place Markovian dynamics on state assignments for each animal, we model separate draws from the prior jointly for each animal, and we share cluster atoms across all regions through use of an HDP.

## 2.3 Infinite mixture of Gaussian processes

### 2.3.1 Gaussian processes and the spectral mixture kernel

For a single window of data, $\boldsymbol{y}_w^{(ar)} \in \mathbb{R}^N$, we wish to model the data in the limit of a continuous-time function (allowing $N \to \infty$), motivating a GP formulation, and we are interested in the spectral properties of the LFP signal in this window. Previous research has established a link between the kernel function of a GP and its spectral properties [10]. We write a distribution over the time-series:

$$y(t) \sim \mathcal{GP}(m(t), k(t, t')), \qquad (4)$$

where $m(t)$ is known as the mean function, and $k(t, t')$ is the covariance function [14]. This framework provides a flexible, structured method to model time-series data. The structure of observations in the output space, $\boldsymbol{y}$, is defined through a careful choice of the covariance function. Since this work aims to model the spectral content of the LFP signal, we set the mean function to 0, and use a recently proposed spectral mixture (SM) kernel [10]. This kernel is defined through a spectral domain representation, $S(s)$, of the stationary kernel, represented by a mixture of $Q$ Gaussian components:

$$\phi(s) = \sum_{q=1}^{Q} \omega_q \mathcal{N}(s; \mu_q, \nu_q), \qquad\qquad S(s) = \frac{1}{2}[\phi(s) + \phi(-s)], \qquad (5)$$

where $\phi(s)$ is reflected about the origin to obtain a valid spectral density, and $\mu_q$, $\nu_q$, and $\omega_q$ respectively define the mean, variance, and relative weight of the $q$-th Gaussian component in the spectral domain. Priors may be placed on these parameters; for example, we use the uninformative priors $\mu_q \sim \text{Uniform}(\mu^{min}, \mu^{max})$, $\nu_q \sim \text{Uniform}(0, \nu^{max})$ and $\omega_q \sim \text{Gamma}(e_0, f_0)$. A bandpass filter is applied to the LFP signal from $\mu^{min}$ to $\mu^{max}$ $Hz$ as a preprocessing step, so this prior knowledge is justified. Also, $\nu^{max}$ is set to prevent overfitting, and $e_0$ and $f_0$ are set to manifest a broad prior.

We assume that only a noisy version of the true function is observed, so the kernel is defined as the Fourier transform of the spectral density $S(s)$ plus white Gaussian noise:

$$f(\tau; \boldsymbol{\theta}) = \sum_{q=1}^{Q} \omega_q \exp\{-2\pi^2 \tau^2 \nu_q\} \cos(2\pi\tau\mu_q), \qquad k(\tau; \boldsymbol{\theta}, \gamma) = f(\tau; \boldsymbol{\theta}) + \gamma^{-1}\delta_\tau, \qquad (6)$$

where the set of parameters $\boldsymbol{\theta} = \{\boldsymbol{\omega}, \boldsymbol{\mu}, \boldsymbol{\nu}\}$ and $\gamma$ define the covariance kernel, $\tau = |t - t'|$, and $\delta_\tau$ is the Kronecker delta function which equals one if $\tau = 0$. We set the prior $\gamma \sim \text{Gamma}(e_1, f_1)$ where the hyperparemeters $e_1$ and $f_1$ are chosen to manifest a broad prior. The formulation of (6) results in an interpretable kernel in the spectral domain, where the weights $\omega_q$ correspond to the relative contribution of each component, the means $\mu_q$ represent spectral peaks, and the variances $\nu_q$ play a role similar to an inverse length-scale.

Through a realization of this Gaussian process, an analytical representation is obtained for the marginal likelihood of the observed data $\boldsymbol{y}$ given the parameters $\{\boldsymbol{\theta}, \gamma\}$, and the observation locations $\boldsymbol{t}$, $p(\boldsymbol{y}|\boldsymbol{\theta}, \gamma, \boldsymbol{t})$. The optimal set of kernel parameters $\{\boldsymbol{\theta}, \gamma\}$ can then be chosen as the set that maximizes the marginal likelihood. Further discussions on the inference for the Gaussian process parameters is presented in Section 3.

### 2.3.2 Generating observed data

To combine the clustering model with our SM kernel, each cluster $\ell$ is associated with a distinct set of kernel parameters $\boldsymbol{\theta}_\ell$. To generate the observations $\{\boldsymbol{y}_w^{(ar)}\}$, where each $\boldsymbol{y}_w^{(ar)} \in \mathbb{R}^N$ has observation times $\boldsymbol{t} = \{t_1, \dots, t_N\}$ such that $|t_i - t_j| = |i - j|\tau$ for all $i$ and $j$, we consider a draw from the multivariate normal distribution:

$$\boldsymbol{y}_w^{(ar)} \sim \mathcal{N}(\boldsymbol{0}, \boldsymbol{\Sigma}_{z_w^{(ar)}}), \qquad\qquad (\boldsymbol{\Sigma}_\ell)_{ij} = k(|t_i - t_j|; \boldsymbol{\theta}_\ell, \gamma), \qquad (7)$$

where each observation is generated from the cluster indicated by $z_w^{(ar)}$ (described in Section 2.2), and each cluster is represented uniquely by a covariance matrix, $\boldsymbol{\Sigma}_\ell$, whose elements are defined through the covariance kernel $k(\tau; \boldsymbol{\theta}_\ell, \gamma)$. Therefore, the parameters $\boldsymbol{\theta}_{z_w^{(ar)}}$ describe the auto-correlation content associated with each $\boldsymbol{y}_w^{(ar)}$.

We address two concerns with this formulation. First, this observation model ignores complex cross-covariance functions between regions. Although LFP measurements exhibit coherence patterns across regions, the generative model in (7) only weakly couples the spectral densities of each region through the brain state. In principle, the generative model could be extended to incorporate this coherence information. Second, (7) does not model the time-series itself as a stochastic process, but rather the preprocessed, 'independent' observation vectors. This shortcoming is not ideal, but the windowing process allows for efficient computation via the mixture of Gaussian processes.

## 3 Inference

In the following, latent model variables are represented by $\boldsymbol{\Omega} = \{\boldsymbol{Z}, \boldsymbol{S}, \boldsymbol{\Phi}, \boldsymbol{\eta}, \boldsymbol{\Lambda}, \boldsymbol{\beta}, \boldsymbol{\Psi}\}$, the kernel parameters to be optimized are $\boldsymbol{\Theta} = \{\{\boldsymbol{\theta}_\ell\}_1^L, \gamma\}$, and $H$ and $L$ are upper limit truncations on the number of brain states and clusters, respectively. As described throughout this section, the proposed algorithm adaptively adjusts the truncation levels on the number of brain states, $H$, and clusters, $L$,

through a series of split-merge moves. The joint probability of the proposed model is

$$p(\boldsymbol{Y}, \boldsymbol{\Omega}, \boldsymbol{\Theta}) = p(\boldsymbol{Y}|\boldsymbol{Z}, \boldsymbol{\Theta})p(\boldsymbol{Z}, \boldsymbol{S}|\boldsymbol{\Phi}, \boldsymbol{\Lambda}, \boldsymbol{\Psi})p(\boldsymbol{\Phi}|\boldsymbol{\eta})p(\boldsymbol{\eta})p(\boldsymbol{\Lambda}|\boldsymbol{\beta})p(\boldsymbol{\beta})p(\boldsymbol{\Psi})p(\boldsymbol{\Theta})$$

$$= \left[ \prod_{a,r,w} p(\boldsymbol{y}_w^{(ar)}|z_w^{(ar)}, \boldsymbol{\Theta})p(z_w^{(ar)}|s_w^{(a)}, \boldsymbol{\Phi}) \right] \left[ p(\boldsymbol{\eta}|\gamma_1) \prod_{r,h} p(\boldsymbol{\phi}_h^{(r)}|\boldsymbol{\eta}, \alpha_1) \right]$$

$$\left[ \prod_a p(s_1^{(a)}|\boldsymbol{\psi}^{(a)})p(\boldsymbol{\psi}^{(a)}) \prod_{w=2}^{W} p(s_w^{(a)}|s_{w-1}^{(a)}, \boldsymbol{\Lambda}^{(a)}) \right] \left[ p(\boldsymbol{\beta}|\gamma_0) \prod_{a,g} p(\boldsymbol{\lambda}_g^{(a)}|\boldsymbol{\beta}, \alpha_0) \right]$$

$$\left[ p(\gamma|e_1, f_1) \prod_{q=1}^{Q} p(\omega_q|e_0, f_0)p(\mu_q|\mu^{min}, \mu^{max})p(\nu_q|\nu^{max}) \right]. \tag{8}$$

A variational inference scheme is developed to update $\boldsymbol{\Omega}$ and $\boldsymbol{\Theta}$.

## 3.1 Variational inference

With variational inference, an approximate variational posterior distribution is sought that is similar to the true posterior distribution, $q(\boldsymbol{\Omega}, \boldsymbol{\Theta}) \approx p(\boldsymbol{\Omega}, \boldsymbol{\Theta}|\boldsymbol{Y})$. This variational posterior is assumed to have a factorization into simpler distributions, where $q(\boldsymbol{\Omega}, \boldsymbol{\Theta}) = q(\boldsymbol{Z})q(\boldsymbol{S})q(\boldsymbol{\Phi})q(\boldsymbol{\eta})q(\boldsymbol{\Lambda})q(\boldsymbol{\beta})q(\boldsymbol{\Psi})q(\boldsymbol{\Theta})$, with further factorization

$$q(\boldsymbol{Z}) = \prod_{a,r,w} \text{Cat}(z_w^{(ar)}; \boldsymbol{\zeta}_w^{(ar)}), \qquad q(\boldsymbol{\Phi}) = \prod_{h,r} \text{Dir}(\boldsymbol{\phi}_h^{(r)}; \boldsymbol{\nu}_h^{(r)}), \qquad q(\boldsymbol{\eta}) = \delta_{\boldsymbol{\eta}^*}(\boldsymbol{\eta}),$$

$$q(\boldsymbol{S}) = \prod_a q(\{s_w^{(a)}\}_{w=1}^W), \qquad q(\boldsymbol{\Lambda}) = \prod_{g,a} \text{Dir}(\boldsymbol{\lambda}_g^{(a)}; \boldsymbol{\kappa}_g^{(a)}), \qquad q(\boldsymbol{\beta}) = \delta_{\boldsymbol{\beta}^*}(\boldsymbol{\beta}),$$

$$q(\boldsymbol{\Psi}) = \prod_a \delta_{\boldsymbol{\psi}^{(a)*}}(\boldsymbol{\psi}^{(a)}), \qquad q(\boldsymbol{\Theta}) = \prod_j \delta_{\boldsymbol{\Theta}_j^*}(\boldsymbol{\Theta}_j), \tag{9}$$

where only necessary sufficient statistics of the latent factors $q(\{s_w^{(a)}\}_{w=1}^W)$ are required, and the approximate posteriors of $\boldsymbol{\eta}$, $\boldsymbol{\beta}$, $\{\boldsymbol{\psi}^{(a)}\}$ and $\{\boldsymbol{\Theta}_j\}$ are represented by point estimates at $\boldsymbol{\eta}^*$, $\boldsymbol{\beta}^*$, $\{\boldsymbol{\psi}^{(a)*}\}$ and $\{\boldsymbol{\Theta}_j^*\}$, respectively.

The degenerate distributions $\delta_{\boldsymbol{\eta}^*}(\boldsymbol{\eta})$ and $\delta_{\boldsymbol{\beta}^*}(\boldsymbol{\beta})$ are described in previous work on variational inference for HDPs [15, 16]. The idea is that the point estimates of the stick-breaking processes simplify the derivation of the variational posterior, and the authors of [16] show that obtaining a full posterior distribution on the stick-breaking weights has little impact on model fitting since the variational lower bound is not heavily influenced by the terms dependent on $\boldsymbol{\eta}$ and $\boldsymbol{\beta}$. Furthermore, the Dirichlet process is truncated for both the number of states and the number of clusters such that $q(z_w^{(ar)} = \ell) = 0$ for $\ell > L$ and $q(s_w^{(a)} = h) = 0$ for $h > H$. This truncation method (see [17] for details) is notably different than other common truncation methods of the DP (e.g., [18] and [19]), and is primarily important for facilitating the split-merge inference techniques described in Section 3.2.

In mean-field variational inference, the variational distribution $q(\boldsymbol{\Omega}, \boldsymbol{\Theta})$ is chosen such that the Kullback-Leibler divergence of $p(\boldsymbol{\Omega}, \boldsymbol{\Theta}|\boldsymbol{Y})$ from $q(\boldsymbol{\Omega}, \boldsymbol{\Theta})$, $D_{\text{KL}}(q(\boldsymbol{\Omega}, \boldsymbol{\Theta})||p(\boldsymbol{\Omega}, \boldsymbol{\Theta}|\boldsymbol{Y}))$, is minimized. This is equivalent to maximizing the evidence lower bound (also known as the variational free energy in the DCM literature), $\mathcal{L}(q) = \mathbb{E}_q[\log p(\boldsymbol{Y}, \boldsymbol{\Omega}, \boldsymbol{\Theta})] - \mathbb{E}_q[\log q(\boldsymbol{\Omega}, \boldsymbol{\Theta})]$, where both expectations are taken with respect to the variational distribution. The resulting lower bound is

$$\mathcal{L}(q) = \mathbb{E}[\ln p(\boldsymbol{Y}|\boldsymbol{Z}, \boldsymbol{\Theta})] + \mathbb{E}[\ln p(\boldsymbol{Z}, \boldsymbol{S}|\boldsymbol{\Phi}, \boldsymbol{\Lambda}, \boldsymbol{\Psi})] + \mathbb{E}[\ln p(\boldsymbol{\Phi}|\boldsymbol{\eta})] + \mathbb{E}[\ln p(\boldsymbol{\eta})] + \mathbb{E}[\ln p(\boldsymbol{\Lambda}|\boldsymbol{\beta})]$$

$$+ \mathbb{E}[\ln p(\boldsymbol{\beta})] + \mathbb{E}[\ln p(\boldsymbol{\Psi})] + \mathbb{E}[\ln p(\boldsymbol{\Theta})] + \mathbb{H}[q(\boldsymbol{Z})] + \mathbb{H}[q(\boldsymbol{S})] + \mathbb{H}[q(\boldsymbol{\Phi})] + \mathbb{H}[q(\boldsymbol{\Lambda})], \tag{10}$$

where all expectations are with respect to the variational distribution, the hyperparameters are excluded for notational simplicity, and we define $\mathbb{H}[q(\cdot)]$ as the sum over the entropies of the individual factors of $q(\cdot)$. Due to the degenerate approximations for $q(\boldsymbol{\eta})$, $q(\boldsymbol{\beta})$, $q(\boldsymbol{\Psi})$ and $q(\boldsymbol{\Theta})$, these full posterior distributions are not obtained, and, therefore, the terms $\mathbb{H}[q(\boldsymbol{\eta})]$, $\mathbb{H}[q(\boldsymbol{\beta})]$, $\mathbb{H}[q(\boldsymbol{\Psi})]$ and $\mathbb{H}[q(\boldsymbol{\Theta})]$ are set to zero in the lower bound.

The updates for $\boldsymbol{\zeta}_w^{(ar)}$ and $\boldsymbol{\nu}_h^{(r)}$ are standard. Variational inference for the HDP-HMM is detailed in other work (e.g., see [20, 21]); using these methods, updates for $\boldsymbol{\kappa}_g^{(a)}$, $\boldsymbol{\psi}^{(a)}$ and the necessary expected sufficient statistics of the factors of $q(\boldsymbol{S})$ are realized. Finally, updates for $\boldsymbol{\beta}^*$, $\boldsymbol{\eta}^*$ and $\{\boldsymbol{\Theta}_j\}$ are non-conjugate, so a gradient-ascent method is performed to optimize these values. We use a simple resilient back-propagation (Rprop), though most line-search methods should suffice. Details on all updates and taking the gradient of $\mathcal{L}(q)$ with respect to $\boldsymbol{\beta}$, $\boldsymbol{\eta}$ and $\{\boldsymbol{\Theta}_j\}$ are found in the Supplemental Material.

## 3.2 Split-merge moves

During inference, a series of split and merge operations are used to help the algorithm jump out of local optima [22]. This work takes the viewpoint that two clusters (or states) should merge only if the variational lower bound increases, and, when a split is proposed for a cluster (or state), it should always be accepted, whether or not the split increases the variational lower bound. If the split is not appropriate, a future merge step is expected to undo this operation. In this way, the opportunity is provided for cluster and state assignments to jump out of local optima, allowing the inference algorithm to readjust assignments as desired.

**Merge states**: To merge states $h'$ and $h''$ into a new state $h$, new parameters are initialized as: $\rho_{wh}^{(a)} = \rho_{wh'}^{(a)} + \rho_{wh''}^{(a)}$, $\kappa_{gh}^{(a)} = \kappa_{gh'}^{(a)} + \kappa_{gh''}^{(a)}$, $\beta_h^* = \beta_{h'}^* + \beta_{h''}^*$, and $v_h^{(a)} = v_{h'}^{(a)} + v_{h''}^{(a)}$, such that the model now has a truncation at $H^{new} = H - 1$ states. In order to account for problems with merging two states in an HMM, a single restricted iteration is allowed, where only the state-dependent variational parameters in $\Omega^{new}$ are updated, producing a new distribution $q(\Omega^{new})$. The merge is accepted (i.e., $\Omega = \Omega^{new}$) if $\mathcal{L}(q(\Omega^{new})) > \mathcal{L}(q(\Omega))$. Since these computations are not excessive, all possible state merges are computed and a small number of merges are accepted per iteration.

**Merge clusters**: To merge clusters $\ell'$ and $\ell''$ into a new cluster $\ell$, new parameters are initialized as: $\zeta_{w\ell}^{(ar)} = \zeta_{w\ell'}^{(ar)} + \zeta_{w\ell''}^{(ar)}$, $\nu_{h\ell}^{(r)} = \nu_{h\ell'}^{(r)} + \nu_{h\ell''}^{(r)}$, $\eta_\ell^* = \eta_{\ell'}^* + \eta_{\ell''}^*$, and $\theta_\ell^{new} = \theta^*$, such that there is a truncation at $L^{new} = L - 1$ clusters. We set $\theta^* = \theta_{\ell'}$ for simplicity, and allow a restricted iteration of updates to $\Omega^{new}$ and $\theta_\ell^{new}$. The merge is accepted (i.e., $\Omega = \Omega^{new}$ and $\Theta = \Theta^{new}$) if the lower bound is improved, $\mathcal{L}(q(\Omega^{new}, \Theta^{new})) > \mathcal{L}(q(\Omega, \Theta))$. Since the restricted iteration for $\theta_\ell^{new}$ is expensive, only a few cluster merges may be proposed at a time. Therefore, merges are proposed for clusters with the smallest earth mover's distance [23] between their spectral densities.

**Split step**: When splitting states and clusters, the opposite process to the initialization of the merging procedures described above is performed. For clusters, data points within a cluster $\ell$ are randomly chosen to stay in cluster $\ell$ or split to a new cluster $\ell'$. For splitting state $h$, the cluster assignment vector $\phi_h^{(r)}$ is replicated and windows within state $h$ are randomly chosen to stay in state $h$ or split to a new cluster $h'$. Regardless of how this effects the lower bound, a split step is always accepted.

For implementation details, we allow the model to accept 3 state merges every third iteration, propose 5 cluster merges every third iteration, and split one state and one cluster every third iteration. Therefore, every iteration may affect the truncation level of either the number of states or clusters. A 'burn-in' period is allowed before starting the proposing of splits/merges, and a 'burn-out' period is employed in which split proposals cease. In this way, the algorithm has guarantees of improving the lower bound only during iterations when a split is not proposed, and convergence tests are only considered during the burn-out period.

## 4 Datasets

Three datasets are considered in this work, as follows:

**Toy data**: Data is generated for a single animal according to the proposed model in Section 2. The purpose of this dataset is to ensure the inference scheme can recover known ground truth, since ground truth information is not known for the real datasets. We set $L = 5$ and $H = 3$. For each cluster, a spectral density was generated with $Q = 4$, $\mu_q \sim \text{Unif}(4, 50)$, $\nu_q \sim \text{Unif}(1, 50)$ and $\omega \sim \text{Dir}(1, \ldots, 1)$. The cluster usage probability vector was drawn $\phi_h^{(r)} \sim \text{Dir}(\frac{1}{10}, \ldots, \frac{1}{10})$. State transition probabilities were drawn according to $\lambda_{gh} \sim \text{Unif}(0, 1) + 10\delta_{(g=h)}$. States were assigned to $W = 1000$ windows according to an HMM with transition matrix $\Lambda$, and cluster assignments were drawn conditioned on this state. Data with $N = 200$ was drawn for each window.

**Sleep data**: Twelve hours of LFP data from sixteen different brain regions were recorded from three mice naturally transitioning through different levels of sleep arousal. Due to the high number of brain regions, we present only three hours of sleep data from a single mouse for simplicity. The multi-animal analysis is reserved for the novel environment dataset.

**Novel environment data**: Thirty minutes of LFP data from five brain regions was recorded from five mice who were moved from their home cage to a novel environment approximately nine minutes into the recording. Placing animals into novel environments has been shown to increase arousal, and

should therefore result in (at least one) network state change [3]. Data acquisition methods for the latter two datasets are discussed in [24].

## 5 Results

For all results, we set $Q = 10$, $H = 15$, $L = 25$, stop the 'burn-in' period after iteration 6, and start the subsequent computation period after iteration 25. Hyperparameters were set to $\gamma_0 = \gamma_1 = .01$, $\alpha_0 = \alpha_1 = 1$, $\mu^{min} = 0$, $\mu^{max} = 50$, $\nu^{max} = 10$, and $e_0 = f_0 = 10^{-6}$. In all results, the model was seen to converge to a local optima after 30 iterations, and each iteration took on the order of 20 seconds using Matlab code on a PC with a 2.30GHz quad-core CPU and 8GB RAM.

Figure 2 shows results on the toy data. The model correctly recovers exactly 3 states and 5 clusters, and, as seen in the figure, the state assignments and spectral densities of each cluster component are recovered almost perfectly. The model was implemented for different values of the noise variance, $\gamma^{-1}$, and, though not shown, in all cases the noise variance was recovered accurately during inference, implying the spectral mixture kernels are not overfitting the noise. In this way, we confirm that the inference scheme recovers a ground truth. For further model verification, ten-fold cross-validation was used to compute predictive probabilities for held-out data (reported in Table 1), where we compare to two simpler versions of our model: 1) the HDP-HMM on brain states in (1) is replaced with an HDP, and 2) a single brain state. For the HDP-HMM, the hold-out data was considered as 'missing data' in the training data and the window index was used to assign time-dependent probabilities over clusters, whereas in the HDP and Single State models it was simply withheld from the training data. We see large predictive performance gains when considering multiple brain states, and even more improvement on average (though modest) when considering an HDP-HMM.

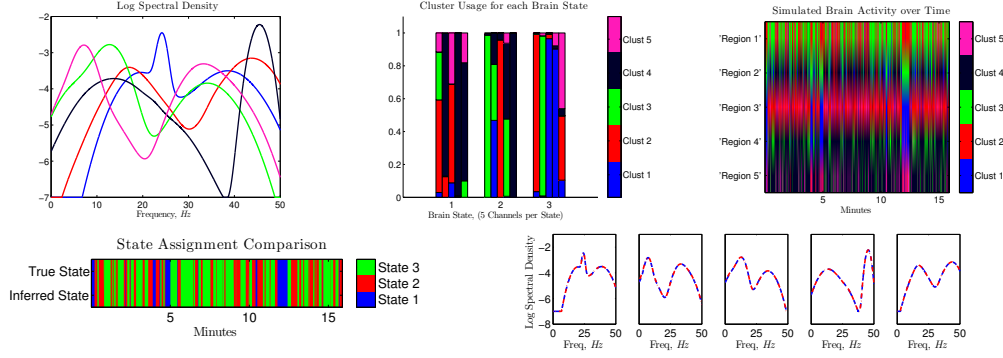

Figure 2: Toy data results. **Top row** shows the generated toy data. From left to right: the five spectral functions, each associated with a component in the mixture model; the probability of each of these five components occurring for all five regions in each brain state; the generated brain state assignments from a 3-state HMM along with the generated cluster assignments for the five simulated regions. The **bottom row** shows the results of our model. On the left, a comparison of the recovered state vs. the true state for all time; on the right, an alignment of the five recovered kernels to the spectral density ground truth.

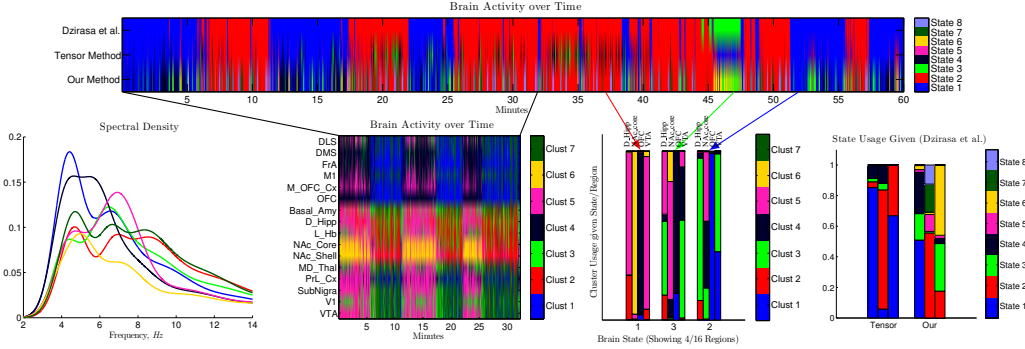

Figure 3: Sleep data results. **Top**: A comparison of brain state assignments from our method to two other methods. **Bottom Left**: Spectral density of the 7 inferred clusters. **Middle Left**: Cluster assignments over time for 16 different brain regions, sorted by similarity. **Middle Right**: Given brain states 1, 2 and 3, we show cluster assignment probabilities for 4 different brain regions: the hippocampus (D_Hipp), nucleus accumbens core (NAc_core), orbitofrontal cortex (OFC) and ventral tegmental area (VTA) from left to right, respectively. **Right**: State assignments of our method and the tensor method conditioned on the method of [6].

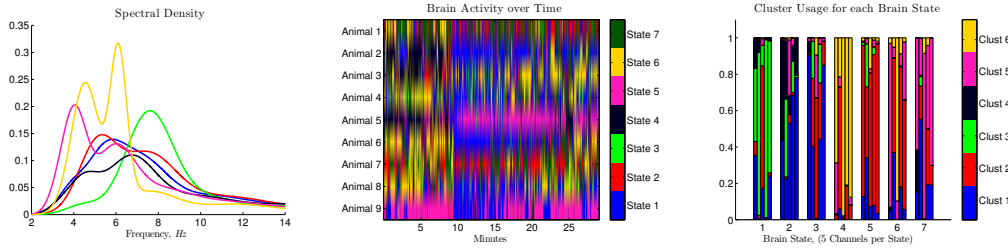

Figure 4: Novel environment data results. **Left**: The log spectral density of the 6 inferred clusters. **Middle**: State assignments for all 9 animals over a 30 minute period. There are 7 inferred states, and each state has a distribution over clusters for each region, as seen on the **right**.

| Dataset | HDP-HMM | HDP | Single State |
|---|---|---|---|
| Toy ($\times 10^5$) | **−1.686 (±0.053)** | −1.688 (±0.053) | −1.718 (±0.054) |
| Sleep ($\times 10^6$) | **−1.677 (±0.030)** | −1.682 (±0.020) | −1.874 (±0.019) |
| Novel ($\times 10^5$) | **−5.932 (±0.040)** | −5.973 (±0.034) | −6.962 (±0.063) |

Table 1: Average held-out log predictive probability for different priors on brain states: HDP-HMM, HDP, and a single state. The data consists of $W$ time-series windows for $R$ regions of $A$ animals; at random, 10% of these time-series windows were held-out, and the predictive distribution was used to determine their likelihood.

The sleep and novel environment results are presented in Figures 3 and 4, respectively. With the sleep dataset, our results are compared with the two methods discussed in the Introduction: that of [6, 7], and the *tensor method* of [8]. We refer to the Supplemental Material for exact specifications of the tensor method.

For each of these datasets, we infer the intended arousal states. In the novel environment data, we observe broad arousal changes at 9–minutes for all animals, as expected. In the sleep data, we successfully uncover *at least* as many states as the simple approach of [6, 7], to include SWS, REM and WK states. Thus far neuroscientists have focused primarily on 2 stages of sleep (NREM and REM), but as many as 5 have been discussed (4 different stages of NREM sleep, and 1 stage of REM). Different stages of sleep affect memory and behavior in different ways (e.g., see [25]), as does the number of times animals transition between these states [26]. Our results suggest that there may be even more levels of sleep that should be considered (e.g., transition states and sub states). This is very interesting and important for neuroscientists to know, because it is possible that each of our newly observed states could affect memory and behavior in different ways. There is no other published method that has provided evidence of these other states.

In addition to brain states, we infer spectral information for each brain region through cluster assignments. Though not the primary focus of this work, it is interesting that groups of brain regions tend to share similar attributes. In Figure 3, we have sorted brain regions into groups based on cluster assignment similarity, essentially recovering a 'network' of the brain. This underscores the power of the proposed method: not only do we develop unsupervised methods to classify whole-brain activity into states, we infer the cross-region/animal relationships within these states.

## 6 Conclusion

The contributions of this paper are three-fold. First, we design an extension of the infinite tensor mixture model, incorporating time dependency. Second, we develop variational inference for the proposed generative model, including an efficient inference scheme using split-merge moves for two general models: the ITM and iHMM. To the authors' knowledge, neither of these inference schemes have been developed previously. Finally, with respect to neuroscience application, we model brain states given multi-channel LFP data in a principled manner, showing significant advantages over other potential approaches to modeling brain states. Using the proposed framework, we discover distinct brain states directly from the raw, filtered data, defined by their spectral content and network properties, and we can infer relationships between and share statistical strength across data from multiple animals.

**Acknowledgments**

The research reported here was funded in part by ARO, DARPA, DOE, NGA and ONR.

# References

[1] C D Gilbert and M Sigman, "Brain States: Top-down Influences in Sensory Processing.," *Neuron*, vol. 54, no. 5, pp. 677–96, June 2007.

[2] A Kohn, A Zandvakili, and M A Smith, "Correlations and Brain States: from Electrophysiology to Functional Imaging.," *Curr. Opin. Neurobiol.*, vol. 19, no. 4, Aug. 2009.

[3] D Pfaff, A Ribeiro, J Matthews, and L Kow, "Concepts and Mechanisms of Generalized Central Nervous System Arousal," *ANYAS*, Jan. 2008.

[4] P J Lang and M M Bradley, "Emotion and the Motivational Brain," *Biol. Psychol.*, vol. 84, no. 3, pp. 437–50, July 2010.

[5] K J Friston, L Harrison, and W Penny, "Dynamic Causal Modelling," *NeuroImage*, vol. 19, no. 4, pp. 1273–1302, 2003.

[6] K Dzirasa, S Ribeiro, R Costa, L M Santos, S C Lin, A Grosmark, T D Sotnikova, R R Gainetdinov, M G Caron, and M A L Nicolelis, "Dopaminergic Control of Sleep–Wake States," *J. Neurosci.*, vol. 26, no. 41, pp. 10577–10589, 2006.

[7] D Gervasoni, S C Lin, S Ribeiro, E S Soares, J Pantoja, and M A L Nicolelis, "Global Forebrain Dynamics Predict Rat Behavioral States and their Transitions," *J. Neurosci.*, vol. 24, no. 49, pp. 11137–11147, 2004.

[8] P Rai, Y Wang, S Guo, G Chen, D Dunson, and L Carin, "Scalable Bayesian Low-Rank Decomposition of Incomplete Multiway Tensors," *ICML*, 2014.

[9] M J Beal, Z Ghahramani, and C E Rasmussen, "The Infinite Hidden Markov Model," *NIPS*, 2002.

[10] A G Wilson and R P Adams, "Gaussian Process Kernels for Pattern Discovery and Extrapolation," *ICML*, 2013.

[11] J Murray and D B Dunson, "Bayesian learning of joint distributions of objects," *AISTATS*, 2013.

[12] Y W Teh, M I Jordan, M J Beal, and D M Blei, "Sharing Clusters Among Related Groups : Hierarchical Dirichlet Processes," *NIPS*, 2005.

[13] R A Harshman, "Foundations of the Parafac Procedure," *Work. Pap. Phonetics*, 1970.

[14] C E Rasmussen and C K I Williams, *Gaussian Processes for Machine Learning*, vol. 14, Apr. 2006.

[15] M Bryant and E B Sudderth, "Truly nonparametric online variational inference for hierarchical Dirichlet processes," *NIPS*, pp. 1–9, 2012.

[16] P Liang, S Petrov, M I Jordan, and D Klein, "The Infinite PCFG using Hierarchical Dirichlet Processes," *Conf. Empir. Methods Nat. Lang. Process. Comput. Nat. Lang. Learn.*, pp. 688–697, 2007.

[17] Y W Teh, K Kurihara, and M Welling, "Collapsed Variational Inference for HDP," *NIPS*, 2007.

[18] D M Blei and M I Jordan, "Variational Inference for Dirichlet Process Mixtures," *Bayesian Anal*, 2004.

[19] K Kurihara, M Welling, and N Vlassis, "Accelerated Variational Dirichlet Process Mixtures," *NIPS*, 2007.

[20] M J Beal, "Variational Algorithms for Approximate Bayesian Inference," *Diss. Univ. London*, 2003.

[21] J Paisley and L Carin, "Hidden Markov Models With Stick-Breaking Priors," *IEEE Trans. Signal Process.*, vol. 57, no. 10, pp. 3905–3917, 2009.

[22] S Jain and R M Neal, "Splitting and merging components of a nonconjugate Dirichlet process mixture model," *Bayesian Anal*, Sept. 2007.

[23] Y Rubner, C Tomasi, and L J Guibas, "The Earth Movers Distance as a Metric for Image Retrieval," *Int. J. Comput. Vis.*, vol. 40, no. 2, pp. 99–121, 2000.

[24] K Dzirasa, R Fuentes, S Kumar, J M Potes, and M A L Nicolelis, "Chronic in Vivo Multi-Circuit Neurophysiological Recordings in Mice," *J. Neurosci. Methods*, vol. 195, no. 1, pp. 36–46, Jan. 2011.

[25] M A Tucker, Y Hirota, E J Wamsley, H Lau, A Chaklader, and W Fishbein, "A Daytime Nap Containing Solely Non-REM Sleep Enhances Declarative but not Procedural Memory.," *Neurobiol. Learn. Mem.*, vol. 86, no. 2, pp. 241–7, Sept. 2006.

[26] A Rolls, D Colas, A Adamantidis, M Carter, T Lanre-Amos, H C Heller, and L de Lecea, "Optogenetic Disruption of Sleep Continuity Impairs Memory Consolidation," *PNAS*, vol. 108, no. 32, pp. 13305–10, Aug. 2011.

